# BLURD: Benchmarking and Learning using a Unified Rendering and Diffusion Model

**Boris Repasky    Anthony Dick    Ehsan Abbasnejad**
Australian Institute for Machine Learning (AIML),
Department of Computer Science, University of Adelaide
`boris.repasky,anthony.dick,ehsan.abbasnejad@adelaide.edu.au`

## Abstract

Recent advancements in pre-trained vision models have made them pivotal in computer vision, emphasizing the need for their thorough evaluation and benchmarking. This evaluation needs to consider various factors of variation, their potential biases, shortcuts, and inaccuracies that ultimately lead to disparate performance in models. Such evaluations are conventionally done using either synthetic data from 2D or 3D rendering software or real-world images in controlled settings. Synthetic methods offer full control and flexibility, while real-world methods are limited by high costs and less adaptability. Moreover, 3D rendering can't yet fully replicate real photography, creating a realism gap. In this paper, we introduce BLURD–Benchmarking and Learning using a Unified Rendering and Diffusion Model–a novel method combining 3D rendering and Stable Diffusion to bridge this gap in representation learning. With BLURD we create a new family of datasets that allow for the creation of both 3D rendered and photo-realistic images with identical factors. BLURD, therefore, provides deeper insights into the representations learned by various CLIP backbones. The source code for creating the BLURD datasets is available at `https://github.com/squaringTheCircle/BLURD`.

## 1   Introduction

Using large pre-trained models such as CLIP [44] has become the standard practice in computer vision. These models are primarily trained on data from the internet without explicit human supervision, making them increasingly powerful. However, evaluating their performance, especially as they are employed in increasingly more critical applications, is crucial. One challenge is the significant data contamination in existing benchmarks. Additionally, evaluating these models on various factors of variation-elements that could create alternative inputs-is important. It enables a better understanding of the unknown underlying biases in the representations learned by these models. Further, it is important to be able to build datasets with known factors of variations that could mitigate these biases via fine-tuning [62].

Many existing datasets do not explicitly consider these factors, but their evaluation is vital. For example, the widespread issue of racial biases stemming from the uneven representation of human faces requires access to datasets that allow for precise manipulation of factors of variation, e.g. gender, hair color, etc [55]. One needs to be able to manipulate one factor (e.g. hair color) while maintaining other variables constant, to be able to generate an unbiased dataset. This is a significant challenge.

Current datasets created for representation learning and used to evaluate and benchmark pre-trained models are commonly created using either synthetic data, generated using approaches like 3D rendering software, e.g. [8], or real-world photographs, captured in controlled settings, e.g. MPI3D [25]. Synthetic methods, favored for their flexibility and lower costs, can utilize existing tools from gaming and 3D modeling [17]. Despite their utility, these methods often fall short in realism, crucially needed

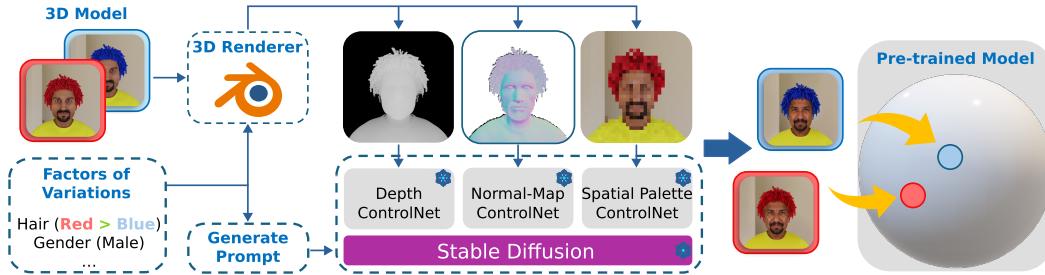

Figure 1: Given the input's factors of variation and its 3D model, our method extracts spatial and visual features. These features, along with textual prompts derived from the input factors, are used to produce photo-realistic outputs. Adjusting any of these factors allows for the generation of alternative outputs that maintain consistency in the unchanged features. In the example here, changing hair color from red to blue in the factors generates alternative prompts and visual features and corresponding outputs.

for the reliable assessment of the underlying properties and behavior of pre-trained models. [6]. Advances in generative techniques, particularly diffusion models [53], have addressed some realism aspects for synthetic data generation but still face significant control challenges. In contrast, real-world data collection remains labor-intensive and costly, hindering scalability [25, 40, 7].

To create scalable and flexible approaches that provide direct control of the factors of variation, we propose a novel method called BLURD, **B**enchmarking and **L**earning using a **U**nified **R**endering and **D**iffusion Model (pronounced "blurred"). Our approach integrates the flexibility and controllability of 3D rendering software, specifically Blender [4], with the impressive realism of Stable Diffusion [41] to create highly realistic and controllable datasets. Figure 1 summarizes our approach.

This integration is achieved by conditioning the diffusion generation on the static image attributes extracted from the rendering engine. We export the essential visual aspects of the images needed to generate controlled outputs and utilize ControlNets [67] to incorporate these attributes into the Stable Diffusion process. By merging deterministic synthetic dataset creation techniques with a data-driven generative approach, BLURD offers a unique solution that balances flexibility and realism at a fraction of the cost associated with real-world data collection. This methodology is simple, cost-effective, scalable, flexible, and provides high controllability.

We leverage our BLURD approach to develop a novel facial dataset that captures factors of variation that are typically challenging to represent. This dataset enables the exploration of relationships between representations as anticipated in the input. For example, altering a person's hair color should only move the features in the representation space in one direction. By utilizing our dataset, we can investigate this property, known as equivariance [6]. Additionally, this dataset allows us to critically compare different architectures trained with the same objectives to understand their unique learning characteristics. Moreover, our approach supports the study of counterfactual and causal reasoning, which necessitates using interventional data to observe the effects of specific variations.

In summary, the contributions of this paper are as follows:

- We introduce a novel approach for creating precisely controlled and scalable datasets cheaply. We unified the deterministic and highly controllable rendering engine with the photo-realism of diffusion models.
- Using our approach, we create a new dataset for benchmarking existing machine-learning models.
- Using our dataset, we can evaluate, compare, and diagnose various CLIP architectures. Our findings reveal a significant gap between representations for 3D images and realistic images. While this might seem trivial, it underscores the necessity of benchmarking with realistic images.
- While one model might exhibit higher average performance on a benchmark such as ImageNet, its equivariance could be lower, suggesting that the complex relationships within the representation space are not fully captured. This underscores the limitation of using class name annotations alone to comprehensively understand models trained with self-supervised and contrastive objectives.
- We evaluate our dataset with a real-world face dataset and show that our approach can be used for data augmentation in supervised learning. We also show that in the absence of manually labeled data, the dataset produced by our approach can be used for training.
- We used human evaluators to independently assess the photo-realism of our approach, finding that that BLURD SD is perceived as significantly more photo-realistic than BLURD 3D.

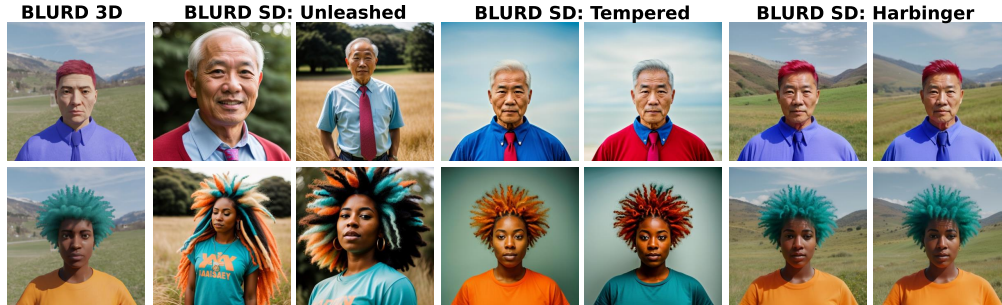

Figure 2: The **BLURD 3D** and **BLURD SD** datasets with ablations. *Left:* the two images on the far left of the figure are examples of **BLURD 3D**. *Middle left:* The four images are examples of **BLURD SD: Unleashed** generated by Stable Difusion conditioned only on text generated from the same factors as the corresponding 3D render. *Middle right:* The next four images are examples of **BLURD SD: Unleashed**, the images generated conditioned on the same text prompt, but also depth and normal maps corresponding to their respective 3D render. *Right:* the last four images are examples of **BLURD SD: Harbinger**, the images are generated by conditioning on text as well as depth and normal maps, the 3d render via img2img and a spatial palette. The figure shows the progression from **BLURD 3D** to **BLURD SD** via a series of ablations, from little constraint to maximal constraint. However, a key feature of **BLURD** is that no level of conditioning sacrifices realism.

## 2 Related Work

**Synthetic Datasets**: Datasets like 3dshapes [8], CLEVR [28], G3DR [45], [61], [3], [56] and [5] are popular examples of diverse and complex visual datasets using 3D rendering engines. However, all these datasets either lack the controllable factors of variation, flexibility, scalability, or realism needed.

**Real-World Datasets**: A large number of the commonly used datasets used for training or benchmarking, such as CelebA [34] and VGGFace2 [34, 10] and most importantly ImageNet [16] leveraged extensive image collections to provide richer, more varied data. MPI3D [24] provides a hybrid approach, offering over a million images of physical 3D objects in a controlled setting, combining realism with experimental controllability. However, these datasets are either collected at significant annotation costs, are highly constrained or require an expensive physical camera apparatus, making them difficult to scale.

**Synthetic Realism and Customizable Environments**: Recent developments have emphasized synthetic realism through advanced 3D rendering technologies and game engines. CARLA [18] and ThreeDWorld [21], for example, utilized customizable simulation environments to support specific research needs, offering a balance between realism and control. Additionally, projects like PUG [6] have leveraged game engines such as Unreal Engine to produce environments that are both highly realistic and flexible, although they still fall short of replicating real-world complexity, require proprietary environments and rely on biased rendering pipelines.

**Facial Recognition Datasets**: In facial recognition, innovative approaches such as GANDiffFace [37] have utilized GANs alongside technologies like Stable Diffusion to create realistic, customizable datasets that also address privacy concerns. These approaches illustrate the potential of data-driven techniques to create photo-realistic synthetic datasets [60, 43]. However, these approaches don't offer the same level of fine control over the factors of variation as 3D environments and rendering techniques.

**Disentanglement Learning**: In recent years, a number of datasets were created for disentanglement learning, where the objective is to learn to decompose complex, high-dimensional data into its underlying factors of variation. This often requires datasets consisting of images generated from a handful of known factors, both for training and evaluation purposes. For instance, [24, 8] are popular examples in spite of their simplicity. However, the need for a high degree of controllability often severely limits the complexity of datasets created for disentanglement learning.

We provide a more in-depth analysis of previous datasets in Appendix C. Furthermore, despite the availability of datasets with varying degrees of granularity, it remains challenging to assess the effectiveness of new, large-scale pre-trained vision models. This paper proposes a methodology for constructing such datasets and outlines criteria for their evaluation to ensure they meet the required properties. Our approach could be easily used for evaluation of multimodal vision-language datasets.

Table 1: **Comparison of BLURD 3D/SD with other comparable datasets.** P: Real photography, S: Sythetically generated, H: Hybrid of real photography and synthetically generated, RT: Real-Time render engine, PT: Pathed-Traced render engine, LP: Low-poly asset, HP: High poly asset, Sprites: 2D bitmap, NT: No textures, LQT: Low resolution textures, HQT: High resolution textures.

| Dataset | Construction | | | Realism | | | | Representation Learning | | |
|---|---|---|---|---|---|---|---|---|---|---|
| | 2D/3D Scene | Photo/Synthetic | Privacy | Render Engine | Assets | Generative | Photoreal Output | Controllable Factors | #Factors | #Images |
| Cars3D | 3D | S | ✓ | RT | LP/LQT | ✗ | ✗ | ✓ | 3 | 19K |
| Sprites | 2D | S | ✓ | RT | Sprites | ✗ | ✗ | ✓ | 8 | 120K |
| dSprites | 2D | S | ✓ | RT | Sprites | ✗ | ✗ | ✓ | 6 | 737K |
| MPI3D | 3D | H | ✓ | RT | LP/LQT | ✗ | ✗ | ✓ | 7 | 1M |
| SmallNORB | 3D | S | ✓ | RT | LP/NT | ✗ | ✗ | ✓ | 5 | 97K |
| Shapes3D | 3D | S | ✓ | RT | LP/LQT | ✗ | ✗ | ✓ | 6 | 480K |
| XYSquares | 2D | S | ✓ | RT | Sprites | ✗ | ✗ | ✓ | 6 | 262K |
| CelebA | 3D | P | ✗ | - | - | - | ✓ | ✗ | 40 | 202K |
| VGGFace2 | 3D | P | ✗ | - | - | - | ✓ | ✗ | 1 | 3.31M |
| PUG | 3D | S | ✓ | RT | HP/HQT | ✗ | ✗ | ✓ | 4 − 7 | 215K |
| **BLURD 3D/SD** | 3D | S | ✓ | PT | HP/HQT | ✓ | ✓ | ✓ | 9/17 | 1.7M |

# 3 Benchmarking and Learning using a Unified Rendering and Diffusion

## 3.1 BLURD Method

Table 1 summarizes the main difference between BLURD 3D/SD and other datasets used for representation learning. We restrict our comparison to only the datasets with a finite set of controllable factors designed for representation learning. However, for completeness we also compare BLURD to two human focused datasets of real photos of human faces. The goal of BLURD is to achieve true photo-realism in a controllable synthetic dataset. Therefore, every design decision was carefully made to maximize realism without sacrificing control. While many datasets use simple objects such as 2D sprites, we constructed a complex 3D scene with high quality 3D assets and textures. Real-time render engines have been used in the past for synthetic representation datasets due to their speed and modern graphics [6], however they introduce systematic error due to their approximate nature. Only physically based path-traced render engines are capable of producing unbiased imagery [4]. Although various researchers have investigated the application of generative models in generating realistic imagery, no approach thus far has attained an acceptable degree of control over the factors of variation until BLURD [37, 22, 1, 59, 2, 50]. Our unique approach of using 3D data generation in unison with generative methods allow BLURD to be fully synthetic, controllable and capable of producing true photo-realism. This unique combination allows for the unlimited generation of images of a particular set of factors cheaply and at scale, without any of the privacy concerns of using real photos. However we do acknowledge that researchers have raised the possibility that some generative models could reproduce their training data, and therefore our method could theoretically produce a real person, we believe this possibility is minor due to our generic prompt template, which does not use any identifiable information and the tight coupling with 3D models that do not correspond to any real persons [54]. Figure 1 provides an overview of the BLURD method.

## 3.2 BLURD Datasets

**BLURD 3D** We present BLURD 3D, a highly controllable synthetic 3D dataset of humans that allows for the alteration of several factors of variation. Each factor in BLURD 3D is accompanied by a path traced 3D render, as well as corresponding depth maps, normal maps, and segmentation masks. To create BLURD 3D we used the path-traced Cycles engine, with high quality assets and plugins from the Blender marketplace [4, 42, 58]. Additionally, BLURD 3D uniquely focuses on human assets, complete with particle hair systems, industry standard high resolution PBR textures and morph targets, providing unprecedented control over creating realisic humans.

By creating a suitable 3D scene and using the Blender Python API, we enable programmatic control over a total of seventeen factors of variations. These factors of variations are age*, gender*, race*, hair style*, beard style*, hair color*, beard color, shirt color*, tie color, eye color, eyeshadow color, eyeliner color, eyebrow color, lipstick color, world background*, world background rotation and camera angle*. We then pre-render and release a subset (denoted with ∗) of the possible factors of

variation as the BLURD 3D dataset. Refer to Appendix B in the supplementary materials for further details of the assets used and their licenses. BLURD 3D itself is released under CC BY-NC 4.0.

**BLURD SD**  Here we present BLURD SD a synthetic photo-realistic dataset for representation learning. The BLURD SD dataset we release varies over the same factors as the BLURD 3D dataset. To create BLURD SD we use a SD 1.5 model fine-tuned on realistic images conditioned on the depth map, normal map, spatial color palette and 3D render generated when creating BLURD 3D [19]. Conditioning by the depth map, normal map, spatial color palette utilize purpose trained ControlNets, while conditioning on the 3D render uses the image-to-image diffusion technique [47, 41, 67].

Among synthetic datasets for representation learning, BLURD SD is unique for merging a deterministic 3D generative process with a data-driven diffusion based generative approach [6, 37, 13, 47, 49, 30, 15, 26, 24, 60, 43]. This distinctive combination enables BLURD SD to attain both controllability and unparalleled photo-realism.

Besides having a controllable yet photo-realistic dataset for representation learning, BLURD SD also allows for greater insight into text-to-image generative models. In particular, we provide three ablations in BLURD SD allowing us to probe how the structure of the representation space is related to common failure modes of diffusion text-to-image models. Prompts that contain several factors of variation will frequently lead to generated images that are hopelessly entangled [41, 47]. Figure 2 has examples where the background, color of the shirt and color of the hair become intertwined, bleeding into one another causing mixed factors in the final image.

The ablations of BLURD SD are denoted BLURD SD: Unleashed, BLURD SD: Tempered and BLURD SD: Harbinger and refer to various weakening of the conditioning on SD. BLURD SD: Unleashed is only conditioned on the prompt, BLURD SD: Tempered is conditioned on the prompt, depth map and normal map and finally BLURD SD: Harbinger utilize the full BLURD method. For brevity we use BLURD SD and BLURD SD: Harbinger interchangeably were context permits, and refer to the individual ablations when discussing experiments. We release BLURD SD under the CC BY-NC 4.0 license. Appendix B in the supplementary materials contains additional details.

**BLURD Mask**  Finally, we present BLURD Mask, a new zero-shot domain adaption dataset and task. We use the rendered segmentation masks to create a new synthetically labeled semantic segmentation dataset together with the hand annotated segmentation masks of the CelebAMask-HQ dataset [32]. To create a more diverse dataset, BLURD Mask samples from the expanded set of seventeen possible factors of variation from BLURD 3D and the additional factors of focal length, lighting strength, and lighting warmth. Then the BLURD pipeline was followed to create the photo-realistic versions of each of the rendered images. The associated masks were combined and unified with the annotations from the CelebAMask-HQ dataset. The resulting dataset contains annotated 3D renders and BLURD photo-realistic images as training sets, with the real world photographs of the CelebAMask-HQ dataset serving as a test set. Figure 7 shows an example of the synthetic images and masks as well as an example of the hand annotated masks from CelebAMask-HQ.

BLURD Mask serves as an ideal benchmark for assessing how well the BLURD method mimics real-world photography. The dataset can be used by future researchers to test their own 3D-to-photorealistic pipelines, allowing for a comprehensive evaluation of their techniques. Furthermore, the dataset can be extended by future researchers with additional factors of variation, enabling the development of even more sophisticated and realistic data-driven rendering pipelines. Full discussion of how BLURD Mask was created can be found in Appendix B. BLURD Mask is similarly released under the CC BY-NC 4.0, however CelebAMask-HQ maintains its original license and remains restricted to non-commercial research and educational purposes only.

## 4   Evaluations on CLIP using the BLURD Dataset

**Limitations and Biases in Pre-trained Models**  The BLURD dataset provides valuable insights into the biases and limitations of current pre-trained CLIP models and their training data. Several insights can be gained by analysing the confusion matrices of our single zero-shot accuracy results. Notably, Figure 3 reveals that all races are often confused with Caucasians in the 3D case, underscoring the pitfalls of relying solely on representation learning datasets that use 3D rendering alone. However, even in BLURD SD: Harbinger CLIP models often had difficulty distinguishing between Caucasians and Hispanic races. Notably, our results show that Laion2b outperformed OpenAI and Laion400m in

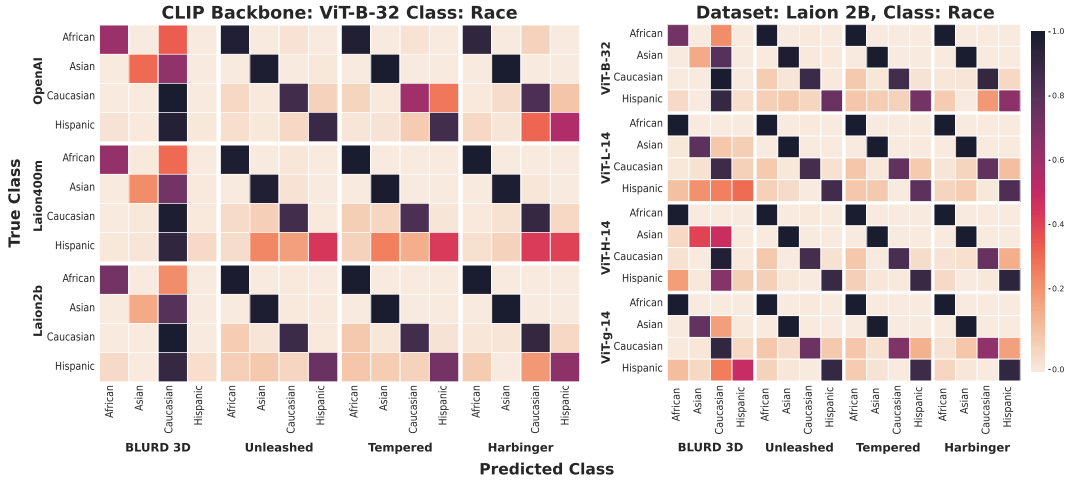

Figure 3: Zero-shot confusion matrices for the factor race. *Left:* We fix the CLIP backbone to ViT-B-32 and provide a confusion matrix for each of the CLIP training datasets OpenAI, Laion400m and Laion2b evaluated for each member of **BLURD**. *Right:* We fix the training dataset to Laion2b and provide a confusion matrix for the CLIP backbones ViT-B-32, ViT-L-14, ViT-H-14-CLIPA and ViT-g-14 evaluated for each member of **BLURD**.

recognizing race, even after controlling for the CLIP model. Furthermore, our findings indicate that the choice of training data plays a significant role in CLIP models' ability to recognize underrepresented groups, with Laion2b demonstrating a notable edge over OpenAI and Laion400m in correctly identifying the race. Additionally, larger models performed better overall both in BLURD 3D and BLURD SD. Outside of race, certain datasets and models showed several shortcomings. The Laion400m dataset has the lowest accuracy determining gender, suggesting an under representation of females. All CLIP models performed poorly at recognizing age, with a tendency to misidentify older individuals. Perhaps most surprisingly, certain models and datasets showed a strong preference for a particular color, with the OpenAI dataset preferring the color green while Laion400M and Laion2B preferred violet. Although our results show that BLURD can provide insight to biases of pre-trained models, it is important to underline the limitations of this analysis. BLURD is not intended for, and should not be used as the sole determinate for whether a machine learning model is free from biases and safe to use in applications where this is a concern.

**Single Factor Zero-shot Classification Accuracy** We evaluate the zero-shot performance of several pre-trained CLIP models trained on various datasets. Experiments were conducted for single-factor zero-shot accuracy by measuring the caption retrieval accuracy on the template "A picture of *factor*." Figure 4 presents aggregated results for the single-factor accuracy experiments over BLURD 3D and the BLURD SD ablations. The results reveal a performance gap in zero-shot accuracy between BLURD 3D and BLURD SD: Harbinger, suggesting that CLIP models and their training datasets are not suited for tasks on 3D images. This may indicate deficiencies in the training data itself, or highlight an inherent deficiency in 3D images in capturing the complexity of a real-world image. The results on the ablations of BLURD SD are also telling; for instance, *tempered* and *unleashed* performed better than *harbinger* on shirt color and tie color, but had worse results on hair color and beard color. A plausible explanation is that the shirt and tie occupy a larger area in the image compared to the hair and beard, which increases the chance of color blending during an unconstrained diffusion process.

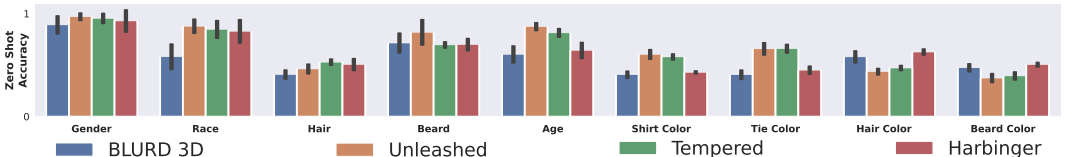

Figure 4: Single factor zero-shot accuracy of CLIP models on the **BLURD** dataset. Here we evaluate the CLIP backbones ViT-B-32 trained on the OpenAI, Laion400m and Laion2b datasets, ViT-L-14 trained on the OpenAI, Laion2b and Datacomp XL datasets and finally ViT-H-14-CLIPA and ViT-g-14 trained on the Laion2b dataset. We show that there exists a significant gap between the renders of **BLURD 3D** and the photo-realistic **BLURD SD** dataset, underlining the need for photo-realism in datasets for representation learning. In particular we note that despite **BLURD 3D** and **BLURD SD:** Harbinger sharing identical underlining factors shifting from 3D to photo-realism improves zero-shot accuracy in almost every class.

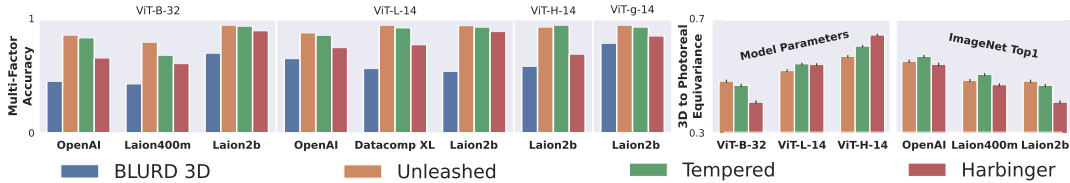

Figure 5: *Left:* Multi-factor zero-shot classification accuracy for each of the CLIP backbones ViT-B-32, ViT-L-14, ViT-H-14-CLIPA and ViT-g-14 trained on the datasets OpenAI, Laion400m, Laion2b and Datacomp XL datasets. The multi-factor zero-shot classification results further reinforce the single-factor zero-shot classification, showing a consistent improvement in the photo-realistic **BLURD SD** datasets over **BLURD 3D**. *Right:* Equivariance of a transformation from 3D render to photo-realism under two scenarios. We analyse both the 3D to photo-realism equivariance of the ViT-B-32 CLIP backbone trained with different datasets and the 3D to photo-realism equivariance across CLIP backbones trained on the Laion2b dataset. We found that the 3D to photo-realism equivariance is negatively correlated with the ImageNet Top-1 score in Vit-B-32, which is closely related to the training dataset size and quality. This suggests that larger training datasets and better zero-shot performance may cause asymmetries between the 3D render and photo-realism representations. However, when we fix the training data and instead vary backbone parameter size, we notice the opposite relationship with larger models having greater symmetry between 3D render and photo-realism representations.

**Multi-Factor Zero-Shot Classification Accuracy** Multi-factor zero-shot classification Accuracy measuring the caption retrieval accuracy of the sentence describing all factors "A picture of a *age race gender*, with *other factors*"". Figure 5 contains the results of multi-factor accuracy. Notably, accuracy is high in the multi-factor case, despite there being a greater number of classes, suggesting better alignment with the full caption compared to the single factor caption. The multi-factor results show some similarities to the single factor results, namely that BLURD SD: Harbinger outperformed BLURD 3D. However *unleashed* and *tempered* outperformed *harbinger*, with *tempered* performing the best. Considering that BLURD SD: Tempered is essentially unconstrained SD 1.5 output the result suggest that despite the visual confusion and mixing of factors evident in *tempered* the image representation and text representation are well aligned. Such a result would suggest that issues Stable Diffusion has with avoiding the mixing of factors in a caption are actually due to the CLIP embedding itself.

**Measuring the Shift from 3D to Photo-realistic** Unique to BLURD is the insight the datasets can provide into how the representation spaces of CLIP models behave undergoing a domain shift from 3D to photo-realistic. More specifically, BLURD allows us to make a factor change in the 3D to photo-realism factor while keeping every other factor constant, providing a measure of the equivariance when undergoing this factor change. Figure 5 shows the relationship between the datasets OpenAI, Laion400m and Laion2b and the 3D render to photo-realism equivariance when evaluated using the ViT-B-32 backbone. Figure 5 shows that equivariance improves with the model parameters, however decreases with the ImageNet Top-1 - a measure of accuracy on ImageNet - holding all else equal. While this analysis is limited by the expense of training large pre-trained models, and hence the small sample size of either having a single CLIP model be trained on multiple datasets or having multiple models trained on the same dataset, these results are still suggestive. Larger models may have greater preservation of the representation space when undergoing a shift from 3D to photo-realistic but counter-intuitively more data may be in fact detrimental.

**Studying the representation space of pre-trained Models with human factors** BLURD 3D and BLURD SD are useful tools in studying the behavior of the region of representation spaces that encode for information related to humans. In particular, studying the equivariance of the factors of variation of BLURD gives insight into which characteristics of humans are captured in a well structured representation and which are disjoint and entangled with other factors. We conducted both text and image equivariance experiments for every factor in BLURD 3D and BLURD SD for a number of CLIP models. See subsection E.1 for the full list of figures. Looking at the Image equivariance Figures 22 23 and 24 we can immediately see that BLURD 3D has very high Equivariance across all factors, with a noticeable drop in all BLURD SD variants. Indicating that the consistent and deterministic nature of 3D rendering make equivariance results difficult to translate directly to the photo-realistic case. Image equivariance tended to be higher for *harbinger* than *unleashed* or *tempered*, however there were notable exception with age and gender, with a particularly low equivariance with ViT-L-14 trained on the Datacomp XL dataset. Text equivariance seen in Figures 25 26 and 27 displayed more mixed results, with BLURD 3D having high text equivariance in beard and camera angle but being outperformed by all BLURD SD ablations in race and gender.

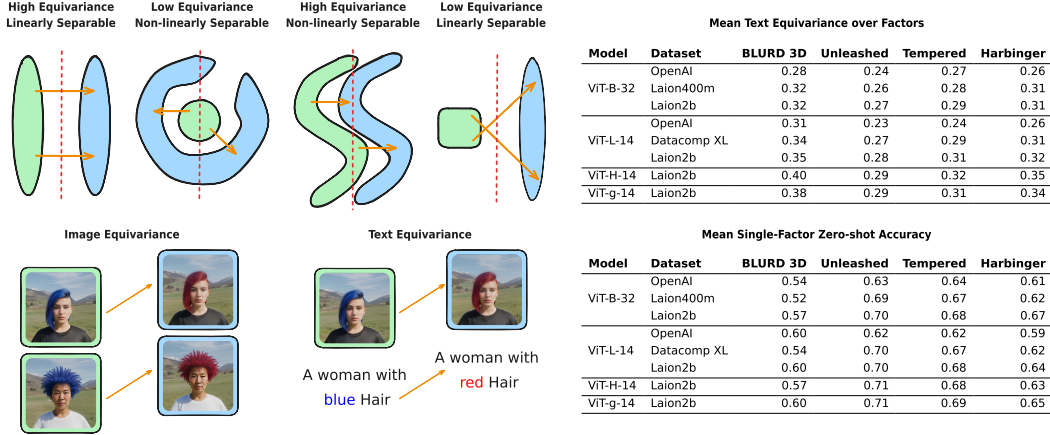

| Model | Dataset | BLURD 3D | Unleashed | Tempered | Harbinger |
|---|---|---|---|---|---|
| | **Mean Text Equivariance over Factors** | | | | |
| | OpenAI | 0.28 | 0.24 | 0.27 | 0.26 |
| ViT-B-32 | Laion400m | 0.32 | 0.26 | 0.28 | 0.31 |
| | Laion2b | 0.32 | 0.27 | 0.29 | 0.31 |
| | OpenAI | 0.31 | 0.23 | 0.24 | 0.26 |
| ViT-L-14 | Datacomp XL | 0.34 | 0.27 | 0.29 | 0.31 |
| | Laion2b | 0.35 | 0.28 | 0.31 | 0.32 |
| ViT-H-14 | Laion2b | 0.40 | 0.29 | 0.32 | 0.35 |
| ViT-g-14 | Laion2b | 0.38 | 0.29 | 0.31 | 0.34 |

| Model | Dataset | BLURD 3D | Unleashed | Tempered | Harbinger |
|---|---|---|---|---|---|
| | **Mean Single-Factor Zero-shot Accuracy** | | | | |
| | OpenAI | 0.54 | 0.63 | 0.64 | 0.61 |
| ViT-B-32 | Laion400m | 0.52 | 0.69 | 0.67 | 0.62 |
| | Laion2b | 0.57 | 0.70 | 0.68 | 0.67 |
| | OpenAI | 0.60 | 0.62 | 0.62 | 0.59 |
| ViT-L-14 | Datacomp XL | 0.54 | 0.70 | 0.67 | 0.62 |
| | Laion2b | 0.60 | 0.70 | 0.68 | 0.64 |
| ViT-H-14 | Laion2b | 0.57 | 0.71 | 0.68 | 0.63 |
| ViT-g-14 | Laion2b | 0.60 | 0.71 | 0.69 | 0.65 |

Figure 6: The interplay between equivariance and linear separability (related to zero-shot accuracy) together with the corresponding **BLURD** Benchmarks. *Top-Left:* Informal diagram of four scenarios of how different equivariance scores and levels of linear separability can co-exist in a representation space. *Bottom-Left:* An example of an image equivariance and a text equivariance calculation, here we show a factor change from blue hair to red hair. The two displacement vectors shown are used to calculate the cosine similarity. Text equivariance compares the displacement vector from a factor change in the caption, with the image displacement vector. Image equivariance compares the displacement vector of two different images undergoing the same factor change. While we use cosine distance to calculate the nearest text embedding for zero-shot accuracy, which better corresponds to separability on the hypersphere, we use linear separability here for simplicity. *Top-Right:* Mean Text Equivariance over Factors which measures the equivariance of two caption-image pairs generated using our approach when we fix all the factors and change only one. *Bottom-Right:* Mean Single-Factor Zero-shot Accuracy which measures the caption retrieval accuracy for the generating factor. Larger values indicate better equivariance and accuracy respectively, and consequently more aligned and separated feature space. For instance, we can observe that the Laion2b training dataset out-performs OpenAI in both mean equivariance and mean single-factor zero-shot accuracy, even across models. This suggest that the Laion2b dataset has superior characteristics

**Equivariance vs. Overall Accuracy** In this section we discuss the insights that can be gained into the representation space of CLIP models by comparing the zero-shot classification accuracy against the image and text equivariances. Figure 6 gives a visual representation of how comparing zero-shot classification accuracy and equivariance can provide details as to the structure of the representation space. Using various CLIP backbones, we can evaluate both accuracy and equivariance. As shown in Figure 4 and Figure 5, we observe surprising differences across backbones. Additionally, comparing the accuracy results in Figure 19 and Figure 20 in the Appendix, we see that certain factors, such as gender have both high accuracy and high equivariance, suggesting that the representation space is well behaved over all the CLIP models. Other factors such as beard and age have much lower equivariance and lower accuracy despite also being binary classes suggesting relatively entangled representation spaces compared to gender. Of particular note is the shirt color factor, as shirt color has relatively poor zero-shot accuracy but very high image and text equivariance on BLURD 3D. While there may be other reasons why zero-shot accuracy on shirt color might be low, one possibility is that there is a mismatch between how separable shirt color is on the CLIP representation spaces compared to how the representation space structure is preserved under a color change. Interestingly, the aforementioned inverse relationship between the zero-shot accuracy of shirt color and hair color across the BLURD datasets is not present in the text or image equivariance. These initial studies show that BLURD is a valuable tool in studying the representation spaces of large pre-trained models.

**Zero Shot Domain Adaptation using BLURD for Segmentation** Domain adaptation, also known as cross-domain adaptation, refers to using a source domain to train a model and applying it to a different but related target domain. In this section we focus on using the BLURD method to improve image segmentation results on real-world datasets when only synthetic 3D datasets are available. We achieve this using BLURD Mask, a fully synthetic photo-realistic semantic segmentation dataset. We show that BLURD Mask can be effectively used as a source domain for learning to segment the CelebAMask-HQ dataset. As the IOU results in Figure 7 demonstrate, attempting to use 3D generated images for training alone results in poor generalizability to real-world imagery. While it is possible to improve generalizability with data augmentation, we show typical semantic segmentation data augmentation operations are insufficient to bridge the 3D render to real-world imagery gap. To

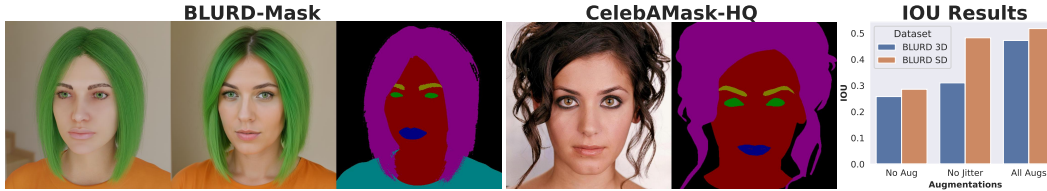

Figure 7: **BLURD Mask** shown together with an example of **CelebAMask-HQ** and with the zero-shot domain adaption results. We conduct the domain adaptation experiment under three levels of data augmentation on the training datasets of **BLURD Mask**, namely **BLURD 3D** and **BLURD SD**. In the first we do not conduct any data augmentation on the input data (*No Aug*), in the second we randomly resize, crop and horizontally flip the input (*No Jitter*), finally in the last experiment we add random brightness and hue shifts to the previous augmentation (*All Augs.*). We demonstrate that using **BLURD** is an effective way to use synthetic 3D images to train on a real-world task.

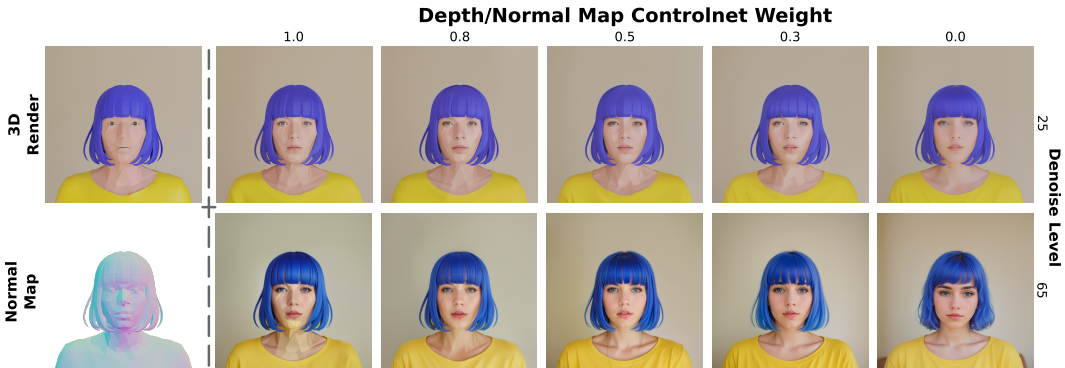

Figure 8: The impact of using low-poly meshes and simple textures. Here we replace the high quality assets used in **BLURD SD** with low quality analogs. We use a low-poly hair mesh together with a decimated human base mesh from Blender Studio, as well as simple single color materials. The figure shows the impact of varying the weighting of both the depth ControlNets and normal map ControlNets from 1.0 to 0.0 at two different denoise levels. The figure demonstrates how the normal and depth ControlNets can be highly sensitive to low-polygon models. However varying the weights of the ControlNets or increasing the denoise level can alleviate this issue. Albiet, with the consequence of reducing the controllability over the desired factors of variation.

demonstrate that effective domain adaptation to the real-world domain requires in domain photo-realism, we trained U-Net models using BLURD 3D and BLURD SD under three data augmentation regimes using the synthetic BLURD Mask segmentation masks [48]. In the first augmentation regime, no data augmentations were applied to the input data. In the second random resizing, cropping, and horizontal flipping augmentation were applied. Finally, random brightness and hue shifts were applied in addition to all of the previous augmentations used. In all three experiments the U-Net trained on the BLURD SD data obtained higher IOU scores when tested on the CelebAMask-HQ test set.

**Impact of Low-Poly Assets on the BLURD SD Pipeline** This section explores the influence of utilizing lower quality 3D models and textures on the BLURD pipeline instead of the higher quality assets used to make the BLURD datasets. To conduct this investigation, we sourced assets under (CC0 and CC-BY) from Blender Studio, MakeHuman, and the community [4, 57], and where appropriate further transformed them into progressively lower-poly versions. Our findings showed that a particle-based hair system was more photo-realistic than a low-poly hair mesh, even at lower denoise levels (see Figure 28). Furthermore, lower resolutions enhanced the appearance of low-poly mesh hair, while the particle-based hair system maintained its quality at higher resolutions (see Figure 29). Notably, both the depth ControlNets and normal map ControlNets were found to be highly sensitive to geometry, leading to visible artifacts when using low-quality assets. However, modifying the weightings of these ControlNets or increasing the denoise level could mitigate this issue, although this came at the cost of reduced control over the desired factors of variation as shown in Figure 8. Overall, these findings highlight the importance of considering the quality and type of 3D assets used in image generation pipelines, and the potential trade-offs that can arise when attempting to improve performance with adjustments to generation parameters. Refer to Appendix F for further details.

**Human Evaluation of the Photo-realism of BLURD**
To provide an independent evaluation of the photo-realism of BLURD SD we conducted a human evaluation study asking participants to assess the perceived photo-realism of a set of 25 images. The study consisted of an online survey where participants were sequentially presented with a series of 25 images. The images were drawn from one of five sets: BLURD 3D, BLURD SD: Unleashed, BLURD SD: Harbinger, pre-selected control images and CelebA-HQ. Each participant was asked to rate the images on a scale of 1 to 5, with 1 being the least photo-realistic and 5 being the most photo-realistic. The five pre-evaluated control images were the same for every participant and appeared

Table 2: Results from the human evaluation study into the perceived photo-realism of BLURD SD. Participants were shown a series of 25 images comprised of images from one of five sets: BLURD 3D, BLURD SD: Unleashed, BLURD SD: Harbinger, pre-selected control images and CelebA-HQ and asked to rate them on a 5-point scale for their perceived photo-realism.

| Dataset | Photo-realism |
|---|---|
| BLURD 3D | 2.42 |
| BLURD SD: Unleashed | 4.19 |
| BLURD SD: Harbinger | 3.98 |
| CelebA-HQ | 4.70 |

at predetermined positions within the 25 total images. Every other image was sampled using a multi-stage sampling method without replacement. Excluding the controls each set contained 1000 possible images that were sampled from five times without replacement. The control images were selected from the internet and consisted of three 3D renders and two real world photographs, all with a central human subject. The results from survey participants who failed to complete the full 25 images or who randomly ranked the control images were discarded. Our findings suggest that BLURD SD: Harbinger is perceived as significantly more photo-realistic than BLURD 3D, while still maintaining the same level of control over the factors of variation. The human evaluation did find that *unleashed* was perceived to be slightly more photo-realistic than *harbinger*, however this may be attributable to the greater variety exhibited in the unconstrained nature of *unleashed*. CelebA-HQ is a dataset of high-quality real celebrity photographs, and served as a benchmark for comparison. CelebA-HQ received the highest ranking for perceived photo-realism, however we note that the images in CelebA-HQ had greater variation in pose, facial expression, background and lighting, as well as the inclusion of recognizable and familiar celebrities. Despite our best efforts these factors may have impacted the results.

## 5    Conclusion

In this paper we have introduced a novel method for creating controllable yet photo-realistic datasets called BLURD. BLURD presents a powerful approach to creating realistic and highly controllable datasets for representation learning. By combining the strengths of 3D rendering and Stable Diffusion, we bridge the realism gap and offer a cost-effective solution for generating photo-realistic images. Additionally, we introduce three new datasets BLURD 3D, BLURD SD and BLURD Mask. The datasets produced using BLURD enable in-depth research on representation spaces and facilitate the study of text-to-image generation failure, domain shifts from 3D to photo-real and biases inherent in large pre-trained models. We further highlight the importance of equivariance and zero-shot accuracy across different factors of variation and explore the relationship between the zero-shot caption retrieval accuracy and equivariance. We introduce a new domain adaptation task using BLURD Mask and show that training on BLURD SD significantly improves IOU results on a real world semantic segmentation dataset when compared to using 3D renders alone. Therefore demonstrating that BLURD can be used for data augmentation and is an effective proxy for true photo-realism. To strengthen this result we conduct a study using human evaluators to assess the perceived photo-realism of the images generated by BLURD and find that BLURD SD is perceived as significantly more photo-realistic than 3D renders alone. Finally, we conduct an ablation study replacing the high-quality assets and textures used in BLURD with low-poly analogs. We find that although BLURD is highly sensitive to the geometry of low-poly assets, causing the generation of artifacts, modifying certain parameters could mitigate this issue, albeit with some trade-offs. Overall we show that BLURD is a flexible, scalable and highly controllable approach to creating photo-realistic datasets for representation learning and other downstream tasks.

## Acknowledgements

This research was funded partially by the Australian Government through the Australian Research Council (Project DP240103278) and was supported by an Australian Government Research Training Program (RTP) Scholarship.

## References

[1] P. Astolfi, A. Casanova, J. Verbeek, P. Vincent, A. Romero-Soriano, and M. Drozdzal. Instance-conditioned gan data augmentation for representation learning, 2023.

[2] S. Azizi, S. Kornblith, C. Saharia, M. Norouzi, and D. J. Fleet. Synthetic data from diffusion models improves imagenet classification, 2023.

[3] S. Bak, P. Carr, and J.-F. Lalonde. Domain adaptation through synthesis for unsupervised person re-identification, 2018. URL `https://arxiv.org/abs/1804.10094`.

[4] Blender Online Community. *Blender - a 3D modelling and rendering package*. Blender Foundation, Stichting Blender Foundation, Amsterdam, 2018. URL `http://www.blender.org`.

[5] J. Bobadilla, A. Guti'errez, R. Yera, and L. F. P. Mart'inez. Creating synthetic datasets for collaborative filtering recommender systems using generative adversarial networks. *Knowl. Based Syst.*, 280:111016, 2023. URL `https://api.semanticscholar.org/CorpusID:257279905`.

[6] F. Bordes, S. Shekhar, M. Ibrahim, D. Bouchacourt, P. Vincent, and A. S. Morcos. PUG: Photorealistic and Semantically Controllable Synthetic Data for Representation Learning, Aug. 2023. URL `http://arxiv.org/abs/2308.03977`. arXiv:2308.03977 [cs].

[7] C. Brown. *The Economics of Data Collection*. University Press, 2019.

[8] C. Burgess and H. Kim. 3d shapes dataset. https://github.com/deepmind/3dshapes-dataset/, 2018.

[9] D. J. Butler, J. Wulff, G. B. Stanley, and M. J. Black. A naturalistic open source movie for optical flow evaluation. In A. Fitzgibbon et al. (Eds.), editor, *European Conf. on Computer Vision (ECCV)*, Part IV, LNCS 7577, pages 611–625. Springer-Verlag, Oct. 2012.

[10] Q. Cao, L. Shen, W. Xie, O. M. Parkhi, and A. Zisserman. Vggface2: A dataset for recognising faces across pose and age, 2018.

[11] P. Cascante-Bonilla, K. Shehada, J. S. Smith, S. Doveh, D. Kim, R. Panda, G. Varol, A. Oliva, V. Ordonez, R. Feris, and L. Karlinsky. Going beyond nouns with vision & language models using synthetic data, 2023.

[12] A. X. Chang, T. Funkhouser, L. Guibas, P. Hanrahan, Q. Huang, Z. Li, S. Savarese, M. Savva, S. Song, H. Su, J. Xiao, L. Yi, and F. Yu. ShapeNet: An Information-Rich 3D Model Repository, Dec. 2015. URL `http://arxiv.org/abs/1512.03012`. arXiv:1512.03012 [cs] version: 1.

[13] Y. Chen, F. Viégas, and M. Wattenberg. Beyond Surface Statistics: Scene Representations in a Latent Diffusion Model, Nov. 2023. URL `http://arxiv.org/abs/2306.05720`. arXiv:2306.05720 [cs].

[14] M. Cherti, R. Beaumont, R. Wightman, M. Wortsman, G. Ilharco, C. Gordon, C. Schuhmann, L. Schmidt, and J. Jitsev. Reproducible scaling laws for contrastive language-image learning. In *Proceedings of the IEEE/CVF Conference on Computer Vision and Pattern Recognition*, pages 2818–2829, 2023.

[15] L. Colbois, T. d. Freitas Pereira, and S. Marcel. On the use of automatically generated synthetic image datasets for benchmarking face recognition. In *2021 IEEE International Joint Conference on Biometrics (IJCB)*, pages 1–8, Aug. 2021. doi: 10.1109/IJCB52358.2021.9484363. URL `https://ieeexplore.ieee.org/abstract/document/9484363`. ISSN: 2474-9699.

[16] J. Deng, W. Dong, R. Socher, L.-J. Li, K. Li, and L. Fei-Fei. Imagenet: A large-scale hierarchical image database. *2009 IEEE Conference on Computer Vision and Pattern Recognition*, pages 248–255, 2009.

[17] J. Doe. The impact of gaming technology on 3d modeling, 2020. URL `https://www.gamingtechjournal.com/3dmodelingimpact`.

[18] A. Dosovitskiy, G. Ros, F. Codevilla, A. Lopez, and V. Koltun. CARLA: An open urban driving simulator. In *Proceedings of the 1st Annual Conference on Robot Learning*, pages 1–16, 2017.

[19] Evgeny. Realistic vision v5.1, 2023. URL `https://huggingface.co/SG161222/Realistic_Vision_V5.1_noVAE`. [Available at `https://huggingface.co/SG161222/Realistic_Vision_V5.1_noVAE`, Online; accessed 16-September-2023].

[20] S. Y. Gadre, G. Ilharco, A. Fang, J. Hayase, G. Smyrnis, T. Nguyen, R. Marten, M. Wortsman, D. Ghosh, J. Zhang, E. Orgad, R. Entezari, G. Daras, S. Pratt, V. Ramanujan, Y. Bitton, K. Marathe, S. Mussmann, R. Vencu, M. Cherti, R. Krishna, P. W. Koh, O. Saukh, A. Ratner, S. Song, H. Hajishirzi, A. Farhadi, R. Beaumont, S. Oh, A. Dimakis, J. Jitsev, Y. Carmon, V. Shankar, and L. Schmidt. Datacomp: In search of the next generation of multimodal datasets. *arXiv preprint arXiv:2304.14108*, 2023.

[21] C. Gan, J. Schwartz, S. Alter, D. Mrowca, M. Schrimpf, J. Traer, J. D. Freitas, J. Kubilius, A. Bhandwaldar, N. Haber, M. Sano, K. Kim, E. Wang, M. Lingelbach, A. Curtis, K. Feigelis, D. M. Bear, D. Gutfreund, D. Cox, A. Torralba, J. J. DiCarlo, J. B. Tenenbaum, J. H. McDermott, and D. L. K. Yamins. Threedworld: A platform for interactive multi-modal physical simulation, 2021.

[22] R. Gandikota, J. Materzynska, J. Fiotto-Kaufman, and D. Bau. Erasing concepts from diffusion models, 2023.

[23] R. Girdhar and D. Ramanan. CATER: A diagnostic dataset for Compositional Actions and TEmporal Reasoning, Apr. 2020. URL `http://arxiv.org/abs/1910.04744`. arXiv:1910.04744 [cs].

[24] M. W. Gondal, M. Wüthrich, Đ. Miladinović, F. Locatello, M. Breidt, V. Volchkov, J. Akpo, O. Bachem, B. Schölkopf, and S. Bauer. On the Transfer of Inductive Bias from Simulation to the Real World: a New Disentanglement Dataset, Nov. 2019. URL `http://arxiv.org/abs/1906.03292`. arXiv:1906.03292 [cs, stat].

[25] M. W. Gondal, M. Wüthrich, Đorđe Miladinović, F. Locatello, M. Breidt, V. Volchkov, J. Akpo, O. Bachem, B. Schölkopf, and S. Bauer. On the transfer of inductive bias from simulation to the real world: a new disentanglement dataset, 2019.

[26] M. Grimmer, H. Zhang, R. Ramachandra, K. Raja, and C. Busch. Generation of Non-Deterministic Synthetic Face Datasets Guided by Identity Priors, Dec. 2021. URL `http://arxiv.org/abs/2112.03632`. arXiv:2112.03632 [cs].

[27] G. Ilharco, M. Wortsman, R. Wightman, C. Gordon, N. Carlini, R. Taori, A. Dave, V. Shankar, H. Namkoong, J. Miller, H. Hajishirzi, A. Farhadi, and L. Schmidt. Openclip, July 2021. URL `https://doi.org/10.5281/zenodo.5143773`.

[28] J. Johnson, B. Hariharan, L. van der Maaten, L. Fei-Fei, C. L. Zitnick, and R. Girshick. Clevr: A diagnostic dataset for compositional language and elementary visual reasoning. In *CVPR*, 2017.

[29] I. Joshi, M. Grimmer, C. Rathgeb, C. Busch, F. Bremond, and A. Dantcheva. Synthetic data in human analysis: A survey, 2022. URL `https://arxiv.org/abs/2208.09191`.

[30] M. Kim, F. Liu, A. Jain, and X. Liu. DCFace: Synthetic Face Generation with Dual Condition Diffusion Model, Apr. 2023. URL `http://arxiv.org/abs/2304.07060`. arXiv:2304.07060 [cs].

[31] Y. LeCun, F. J. Huang, and L. Bottou. Learning methods for generic object recognition with invariance to pose and lighting. In *Proceedings of the 2004 IEEE Computer Society Conference on Computer Vision and Pattern Recognition, 2004. CVPR 2004.*, volume 2, pages II–104 Vol.2, June 2004. doi: 10.1109/CVPR.2004.1315150. URL `https://ieeexplore.ieee.org/document/1315150`. ISSN: 1063-6919.

[32] C.-H. Lee, Z. Liu, L. Wu, and P. Luo. Maskgan: Towards diverse and interactive facial image manipulation. In *IEEE Conference on Computer Vision and Pattern Recognition (CVPR)*, 2020.

[33] Y. Li and S. Mandt. Disentangled Sequential Autoencoder, June 2018. URL `http://arxiv.org/abs/1803.02991`. arXiv:1803.02991 [cs].

[34] Z. Liu, P. Luo, X. Wang, and X. Tang. Deep Learning Face Attributes in the Wild, Sept. 2015. URL `http://arxiv.org/abs/1411.7766`. arXiv:1411.7766 [cs].

[35] S. Madan, T. Henry, J. Dozier, H. Ho, N. Bhandari, T. Sasaki, F. Durand, H. Pfister, and X. Boix. When and how CNNs generalize to out-of-distribution category-viewpoint combinations, Nov. 2021. URL `http://arxiv.org/abs/2007.08032`. arXiv:2007.08032 [cs] version: 3.

[36] L. Matthey, I. Higgins, D. Hassabis, and A. Lerchner. dsprites: Disentanglement testing sprites dataset. https://github.com/deepmind/dsprites-dataset/, 2017.

[37] P. Melzi, C. Rathgeb, R. Tolosana, R. Vera-Rodriguez, D. Lawatsch, F. Domin, and M. Schaubert. GANDiffFace: Controllable Generation of Synthetic Datasets for Face Recognition with Realistic Variations, May 2023. URL `http://arxiv.org/abs/2305.19962`. arXiv:2305.19962 [cs].

[38] N. Michlo, R. Klein, and S. James. Overlooked Implications of the Reconstruction Loss for VAE Disentanglement, Feb. 2022. URL `http://arxiv.org/abs/2202.13341`. arXiv:2202.13341 [cs] version: 1.

[39] C. Mou, X. Wang, L. Xie, Y. Wu, J. Zhang, Z. Qi, Y. Shan, and X. Qie. T2i-adapter: Learning adapters to dig out more controllable ability for text-to-image diffusion models. *arXiv preprint arXiv:2302.08453*, 2023.

[40] D. Pan, L. Zhuo, J. Piao, H. Luo, W. Cheng, Y. Wang, S. Fan, S. Liu, L. Yang, B. Dai, Z. Liu, C. C. Loy, C. Qian, W. Wu, D. Lin, and K.-Y. Lin. Renderme-360: A large digital asset library and benchmarks towards high-fidelity head avatars, 2023.

[41] D. Podell, Z. English, K. Lacey, A. Blattmann, T. Dockhorn, J. Müller, J. Penna, and R. Rombach. SDXL: Improving Latent Diffusion Models for High-Resolution Image Synthesis, July 2023. URL `http://arxiv.org/abs/2307.01952`. arXiv:2307.01952 [cs].

[42] O. J. Post. Human generator, 2023. URL `https://www.humgen3d.com`. [Available at `https://www.humgen3d.com`, Online; accessed 10-October-2023].

[43] H. Qiu, B. Yu, D. Gong, Z. Li, W. Liu, and D. Tao. SynFace: Face Recognition with Synthetic Data, Dec. 2021. URL `http://arxiv.org/abs/2108.07960`. arXiv:2108.07960 [cs].

[44] A. Radford, J. W. Kim, C. Hallacy, A. Ramesh, G. Goh, S. Agarwal, G. Sastry, A. Askell, P. Mishkin, J. Clark, G. Krueger, and I. Sutskever. Learning transferable visual models from natural language supervision. In *ICML*, 2021.

[45] P. Reddy, I. Elezi, and J. Deng. G3dr: Generative 3d reconstruction in imagenet, 2024.

[46] S. E. Reed, Y. Zhang, Y. Zhang, and H. Lee. Deep Visual Analogy-Making. In *Advances in Neural Information Processing Systems*, volume 28. Curran Associates, Inc., 2015. URL `https://papers.nips.cc/paper_files/paper/2015/hash/e07413354875be01a996dc560274708e-Abstract.html`.

[47] R. Rombach, A. Blattmann, D. Lorenz, P. Esser, and B. Ommer. High-Resolution Image Synthesis with Latent Diffusion Models, Apr. 2022. URL `http://arxiv.org/abs/2112.10752`. arXiv:2112.10752 [cs].

[48] O. Ronneberger, P. Fischer, and T. Brox. U-net: Convolutional networks for biomedical image segmentation, 2015.

[49] N. Ruiz, Y. Li, V. Jampani, Y. Pritch, M. Rubinstein, and K. Aberman. DreamBooth: Fine Tuning Text-to-Image Diffusion Models for Subject-Driven Generation, Mar. 2023. URL http://arxiv.org/abs/2208.12242. arXiv:2208.12242 [cs].

[50] M. B. Sariyildiz, K. Alahari, D. Larlus, and Y. Kalantidis. Fake it till you make it: Learning transferable representations from synthetic imagenet clones, 2023.

[51] C. Schuhmann, R. Vencu, R. Beaumont, R. Kaczmarczyk, C. Mullis, A. Katta, T. Coombes, J. Jitsev, and A. Komatsuzaki. LAION-400M: Open Dataset of CLIP-Filtered 400 Million Image-Text Pairs, Nov. 2021. URL http://arxiv.org/abs/2111.02114. arXiv:2111.02114 [cs].

[52] C. Schuhmann, R. Beaumont, R. Vencu, C. W. Gordon, R. Wightman, M. Cherti, T. Coombes, A. Katta, C. Mullis, M. Wortsman, P. Schramowski, S. R. Kundurthy, K. Crowson, L. Schmidt, R. Kaczmarczyk, and J. Jitsev. LAION-5b: An open large-scale dataset for training next generation image-text models. In *Thirty-sixth Conference on Neural Information Processing Systems Datasets and Benchmarks Track*, 2022. URL https://openreview.net/forum?id=M3Y74vmsMcY.

[53] J. Sohl-Dickstein, E. Weiss, N. Maheswaranathan, and S. Ganguli. Denoising diffusion probabilistic models. *Advances in Neural Information Processing Systems*, 28, 2015.

[54] G. Somepalli, V. Singla, M. Goldblum, J. Geiping, and T. Goldstein. Diffusion art or digital forgery? investigating data replication in diffusion models, 2022.

[55] R. Steed and A. Caliskan. Image representations learned with unsupervised pre-training contain human-like biases. In *Proceedings of the 2021 ACM conference on fairness, accountability, and transparency*, pages 701–713, 2021.

[56] X. Sun and L. Zheng. Dissecting person re-identification from the viewpoint of viewpoint, 2019. URL https://arxiv.org/abs/1812.02162.

[57] The MakeHuman team. *MakeHuman - Open source tool for making 3D characters*. MakeHuman Community, 2024. URL http://www.makehumancommunity.org.

[58] C. to Poly Haven. Poly haven: The public 3d asset library, 2023. URL https://polyhaven.com/. [Available at https://polyhaven.com/, Online; accessed 12-October-2023].

[59] B. Trabucco, K. Doherty, M. Gurinas, and R. Salakhutdinov. Effective data augmentation with diffusion models, 2023.

[60] D. S. Trigueros, L. Meng, and M. Hartnett. Generating Photo-Realistic Training Data to Improve Face Recognition Accuracy, Oct. 2018. URL http://arxiv.org/abs/1811.00112. arXiv:1811.00112 [cs, stat].

[61] V. Voleti. Conditional generative modeling for images, 3d animations, and video, 2023.

[62] A. Wang and O. Russakovsky. Overwriting pretrained bias with finetuning data. In *2023 IEEE/CVF International Conference on Computer Vision (ICCV)*, pages 3934–3945, Los Alamitos, CA, USA, oct 2023. IEEE Computer Society. doi: 10.1109/ICCV51070.2023.00366. URL https://doi.ieeecomputersociety.org/10.1109/ICCV51070.2023.00366.

[63] T. Wolf, L. Debut, V. Sanh, J. Chaumond, C. Delangue, A. Moi, P. Cistac, T. Rault, R. Louf, M. Funtowicz, J. Davison, S. Shleifer, P. von Platen, C. Ma, Y. Jernite, J. Plu, C. Xu, T. L. Scao, S. Gugger, M. Drame, Q. Lhoest, and A. M. Rush. HuggingFace's Transformers: State-of-the-art Natural Language Processing, July 2020. URL http://arxiv.org/abs/1910.03771. arXiv:1910.03771 [cs].

[64] J. Wu, L. Wang, B. Yang, F. Li, C. Liu, and J. Zhou. DEFT: Distilling Entangled Factors by Preventing Information Diffusion. *Machine Learning*, 111(6):2275–2295, June 2022. ISSN 0885-6125, 1573-0565. doi: 10.1007/s10994-022-06134-7. URL http://arxiv.org/abs/2102.03986. arXiv:2102.03986 [cs].

[65] J. Wulff, D. J. Butler, G. B. Stanley, and M. J. Black. Lessons and insights from creating a synthetic optical flow benchmark. In A. Fusiello et al. (Eds.), editor, *ECCV Workshop on Unsolved Problems in Optical Flow and Stereo Estimation*, Part II, LNCS 7584, pages 168–177. Springer-Verlag, Oct. 2012.

[66] H. Ye, J. Zhang, S. Liu, X. Han, and W. Yang. Ip-adapter: Text compatible image prompt adapter for text-to-image diffusion models, 2023.

[67] L. Zhang, A. Rao, and M. Agrawala. Adding Conditional Control to Text-to-Image Diffusion Models, Sept. 2023. URL `http://arxiv.org/abs/2302.05543`. arXiv:2302.05543 [cs].

